# Propagation Algorithms for Variational Bayesian Learning

**Zoubin Ghahramani** and **Matthew J. Beal**

Gatsby Computational Neuroscience Unit
University College London
17 Queen Square, London WC1N 3AR, England
{zoubin,m.beal}@gatsby.ucl.ac.uk

## Abstract

Variational approximations are becoming a widespread tool for Bayesian learning of graphical models. We provide some theoretical results for the variational updates in a very general family of *conjugate-exponential* graphical models. We show how the belief propagation and the junction tree algorithms can be used in the inference step of variational Bayesian learning. Applying these results to the Bayesian analysis of linear-Gaussian state-space models we obtain a learning procedure that exploits the Kalman smoothing propagation, while integrating over all model parameters. We demonstrate how this can be used to infer the hidden state dimensionality of the state-space model in a variety of synthetic problems and one real high-dimensional data set.

## 1 Introduction

Bayesian approaches to machine learning have several desirable properties. Bayesian integration does not suffer overfitting (since nothing is *fit* to the data). Prior knowledge can be incorporated naturally and all uncertainty is manipulated in a consistent manner. Moreover it is possible to learn model structures and readily compare between model classes. Unfortunately, for most models of interest a full Bayesian analysis is computationally intractable.

Until recently, approximate approaches to the intractable Bayesian learning problem had relied either on Markov chain Monte Carlo (MCMC) sampling, the Laplace approximation (Gaussian integration), or asymptotic penalties like BIC. The recent introduction of variational methods for Bayesian learning has resulted in the series of papers showing that these methods can be used to rapidly learn the model structure and approximate the evidence in a wide variety of models. In this paper we will not motivate advantages of the variational Bayesian approach as this is done in previous papers [1, 5]. Rather we focus on deriving variational Bayesian (VB) learning in a very general form, relating it to EM, motivating parameter-hidden variable factorisations, and the use of conjugate priors (section 3). We then present several theoretical results relating VB learning to the belief propagation and junction tree algorithms for inference in belief networks and Markov networks (section 4). Finally, we show how these results can be applied to learning the dimensionality of the hidden state space of linear dynamical systems (section 5).

## 2  Variational Bayesian Learning

The basic idea of variational Bayesian learning is to simultaneously approximate the intractable joint distribution over both hidden states and parameters with a simpler distribution, usually by assuming the hidden states and parameters are independent; the log evidence is lower bounded by applying Jensen's inequality twice:

$$
\ln P(\mathbf{y}|\mathcal{M}) \geq \int d\boldsymbol{\theta}\, Q_{\boldsymbol{\theta}}(\boldsymbol{\theta}) \left[ \int d\mathbf{x}\, Q_{\mathbf{x}}(\mathbf{x}) \ln \frac{P(\mathbf{x},\mathbf{y}|\boldsymbol{\theta},\mathcal{M})}{Q_{\mathbf{x}}(\mathbf{x})} + \ln \frac{P(\boldsymbol{\theta}|\mathcal{M})}{Q_{\boldsymbol{\theta}}(\boldsymbol{\theta})} \right] \quad (1)
$$
$$
= \mathcal{F}(Q_{\boldsymbol{\theta}}(\boldsymbol{\theta}), Q_{\mathbf{x}}(\mathbf{x}), \mathbf{y})
$$

where $\mathbf{y}$, $\mathbf{x}$, $\boldsymbol{\theta}$ and $\mathcal{M}$, are observed data, hidden variables, parameters and model class, respectively; $P(\boldsymbol{\theta}|\mathcal{M})$ is a parameter prior under model class $\mathcal{M}$. The lower bound $\mathcal{F}$ is iteratively maximised as a functional of the two free distributions, $Q_{\mathbf{x}}(\mathbf{x})$ and $Q_{\boldsymbol{\theta}}(\boldsymbol{\theta})$. From (1) we can see that this maximisation is equivalent to minimising the KL divergence between $Q_{\mathbf{x}}(\mathbf{x})Q_{\boldsymbol{\theta}}(\boldsymbol{\theta})$ and the joint posterior over hidden states and parameters $P(\mathbf{x},\boldsymbol{\theta}|\mathbf{y},\mathcal{M})$.

This approach was first proposed for one-hidden layer neural networks [6] under the restriction that $Q_{\boldsymbol{\theta}}(\boldsymbol{\theta})$ is Gaussian. It has since been extended to models with hidden variables and the restrictions on $Q_{\boldsymbol{\theta}}(\boldsymbol{\theta})$ and $Q_{\mathbf{x}}(\mathbf{x})$ have been removed in certain models to allow arbitrary distributions [11, 8, 3, 1, 5]. Free-form optimisation with respect to the distributions $Q_{\boldsymbol{\theta}}(\boldsymbol{\theta})$ and $Q_{\mathbf{x}}(\mathbf{x})$ is done using calculus of variations, often resulting in algorithms that appear closely related to the corresponding EM algorithm. We formalise this relationship and others in the following sections.

## 3  Conjugate-Exponential Models

We consider variational Bayesian learning in models that satisfy two conditions:

**Condition (1).** *The complete data likelihood is in the exponential family:*

$$
P(\mathbf{x},\mathbf{y}|\boldsymbol{\theta}) = f(\mathbf{x},\mathbf{y})\, g(\boldsymbol{\theta}) \exp\left\{\boldsymbol{\phi}(\boldsymbol{\theta})^{\top}\mathbf{u}(\mathbf{x},\mathbf{y})\right\}
$$

*where $\boldsymbol{\phi}(\boldsymbol{\theta})$ is the vector of natural parameters, and $\mathbf{u}$ and $f$ and $g$ are the functions that define the exponential family.*

The list of latent-variable models of practical interest with complete-data likelihoods in the exponential family is very long. We mention a few: Gaussian mixtures, factor analysis, hidden Markov models and extensions, switching state-space models, Boltzmann machines, and discrete-variable belief networks.[1] Of course, there are also many as yet undreamed-of models combining Gaussian, Gamma, Poisson, Dirichlet, Wishart, Multinomial, and other distributions.

**Condition (2).** *The parameter prior is conjugate to the complete data likelihood:*

$$
P(\boldsymbol{\theta}|\eta,\boldsymbol{\nu}) = h(\eta,\boldsymbol{\nu})\, g(\boldsymbol{\theta})^{\eta} \exp\left\{\boldsymbol{\phi}(\boldsymbol{\theta})^{\top}\boldsymbol{\nu}\right\}
$$

*where $\eta$ and $\boldsymbol{\nu}$ are hyperparameters of the prior.*

Condition (2) in fact usually implies condition (1). Apart from some irregular cases, it has been shown that the exponential families are the only classes of distributions with a fixed number of sufficient statistics, hence allowing them to have natural conjugate priors. From the definition of conjugacy it is easy to see that the hyperparameters of a conjugate prior can be interpreted as the number ($\eta$) and values ($\boldsymbol{\nu}$) of pseudo-observations under the corresponding likelihood. We call models that satisfy conditions (1) and (2) *conjugate-exponential*.

In Bayesian inference we want to determine the posterior over parameters and hidden variables $P(\mathbf{x}, \boldsymbol{\theta} | \mathbf{y}, \eta, \boldsymbol{\nu})$. In general this posterior is *neither* conjugate nor in the exponential family. We therefore approximate the true posterior by the following factorised distribution: $P(\mathbf{x}, \boldsymbol{\theta} | \mathbf{y}, \eta, \boldsymbol{\nu}) \approx Q(\mathbf{x}, \boldsymbol{\theta}) = Q_{\mathbf{x}}(\mathbf{x}) Q_{\boldsymbol{\theta}}(\boldsymbol{\theta})$, and minimise

$$KL(Q\|P) = \int d\mathbf{x}\, d\boldsymbol{\theta}\; Q(\mathbf{x}, \boldsymbol{\theta}) \ln \frac{Q(\mathbf{x}, \boldsymbol{\theta})}{P(\mathbf{x}, \boldsymbol{\theta} | \mathbf{y}, \eta, \boldsymbol{\nu})}$$

which is equivalent to maximising $\mathcal{F}(Q_{\mathbf{x}}(\mathbf{x}), Q_{\boldsymbol{\theta}}(\boldsymbol{\theta}), \mathbf{y})$. We provide several general results with no proof (the proofs follow from the definitions and Gibbs inequality).

**Theorem 1** *Given an iid data set* $\mathbf{y} = (\mathbf{y}_1, \ldots \mathbf{y}_n)$, *if the model satisfies conditions (1) and (2), then at the maxima of* $\mathcal{F}(Q, \mathbf{y})$ *(minima of* $KL(Q\|P)$*):*

*(a)* $Q_{\boldsymbol{\theta}}(\boldsymbol{\theta})$ *is conjugate and of the form:*

$$Q_{\boldsymbol{\theta}}(\boldsymbol{\theta}) = h(\tilde{\eta}, \tilde{\boldsymbol{\nu}}) g(\boldsymbol{\theta})^{\tilde{\eta}} \exp\left\{ \phi(\boldsymbol{\theta})^{\top} \tilde{\boldsymbol{\nu}} \right\}$$

*where* $\tilde{\eta} = \eta + n$, $\tilde{\boldsymbol{\nu}} = \boldsymbol{\nu} + \sum_{i=1}^{n} \overline{\mathbf{u}}(\mathbf{y}_i)$, *and* $\overline{\mathbf{u}}(\mathbf{y}_i) = \langle \mathbf{u}(\mathbf{x}_i, \mathbf{y}_i) \rangle_Q$, *using* $\langle \cdot \rangle_Q$ *to denote expectation under* $Q$.

*(b)* $Q_{\mathbf{x}}(\mathbf{x}) = \prod_{i=1}^{n} Q_{\mathbf{x}_i}(\mathbf{x}_i)$ *and* $Q_{\mathbf{x}_i}(\mathbf{x}_i)$ *is of the same form as the known parameter posterior:*

$$Q_{\mathbf{x}_i}(\mathbf{x}_i) \quad \propto \quad f(\mathbf{x}_i, \mathbf{y}_i) \exp\left\{ \overline{\phi}(\boldsymbol{\theta})^{\top} \mathbf{u}(\mathbf{x}_i, \mathbf{y}_i) \right\} = P(\mathbf{x}_i | \mathbf{y}_i, \overline{\phi}(\boldsymbol{\theta}))$$

*where* $\overline{\phi}(\boldsymbol{\theta}) = \langle \phi(\boldsymbol{\theta}) \rangle_Q$.

Since $Q_{\boldsymbol{\theta}}(\boldsymbol{\theta})$ and $Q_{\mathbf{x}_i}(\mathbf{x}_i)$ are coupled, (a) and (b) do not provide an analytic solution to the minimisation problem. We therefore solve the optimisation problem numerically by iterating between the fixed point equations given by (a) and (b), and we obtain the following variational Bayesian generalisation of the EM algorithm:

> **VE Step:** Compute the expected sufficient statistics $\mathbf{t}(\mathbf{y}) = \sum_i \overline{\mathbf{u}}(\mathbf{y}_i)$ under the hidden variable distributions $Q_{\mathbf{x}_i}(\mathbf{x}_i)$.

> **VM Step:** Compute the expected natural parameters $\overline{\phi}(\boldsymbol{\theta})$ under the parameter distribution given by $\tilde{\eta}$ and $\tilde{\boldsymbol{\nu}}$.

This reduces to the EM algorithm if we restrict the parameter density to a point estimate (i.e. Dirac delta function), $Q_{\boldsymbol{\theta}}(\boldsymbol{\theta}) = \delta(\boldsymbol{\theta} - \boldsymbol{\theta}^*)$, in which case the M step involves re-estimating $\boldsymbol{\theta}^*$.

Note that unless we make the assumption that the parameters and hidden variables factorise, we will not generally obtain the further hidden variable factorisation over $n$ in (b). In that case, the distributions of $\mathbf{x}_i$ and $\mathbf{x}_j$ will be coupled for all cases $i, j$ in the data set, greatly increasing the overall computational complexity of inference.

## 4   Belief Networks and Markov Networks

The above result can be used to derive variational Bayesian learning algorithms for exponential family distributions that fall into two important special classes.[2]

**Corollary 1:   Conjugate-Exponential Belief Networks.** *Let* $\mathcal{M}$ *be a conjugate-exponential model with hidden and visible variables* $\mathbf{z} = (\mathbf{x}, \mathbf{y})$ *that satisfy a belief network factorisation. That is, each variable* $z_j$ *has parents* $\mathbf{z}_{p_j}$ *and* $P(\mathbf{z} | \boldsymbol{\theta}) = \prod_j P(z_j | \mathbf{z}_{p_j}, \boldsymbol{\theta})$. *Then the approximating joint distribution for* $\mathcal{M}$ *satisfies the same belief network factorisation:*

$$Q_{\mathbf{z}}(\mathbf{z}) = \prod_j Q(z_j | \mathbf{z}_{p_j}, \tilde{\boldsymbol{\theta}})$$

*where the conditional distributions have exactly the same form as those in the original model but with natural parameters $\phi(\tilde{\boldsymbol{\theta}}) = \overline{\phi}(\boldsymbol{\theta})$. Furthermore, with the modified parameters $\tilde{\boldsymbol{\theta}}$, the expectations under the approximating posterior $Q_{\mathbf{x}}(\mathbf{x}) \propto Q_{\mathbf{z}}(\mathbf{z})$ required for the VE Step can be obtained by applying the **belief propagation** algorithm if the network is singly connected and the **junction tree** algorithm if the network is multiply-connected.*

This result is somewhat surprising as it shows that it is possible to infer the hidden states tractably while integrating over an ensemble of model parameters. This result generalises the derivation of variational learning for HMMs in [8], which uses the forward-backward algorithm as a subroutine.

**Theorem 2: Markov Networks.** *Let $\mathcal{M}$ be a model with hidden and visible variables $\mathbf{z} = (\mathbf{x}, \mathbf{y})$ that satisfy a Markov network factorisation. That is, the joint density can be written as a product of clique-potentials $\psi_j$, $P(\mathbf{z}|\boldsymbol{\theta}) = g(\boldsymbol{\theta}) \prod_j \psi_j(C_j, \boldsymbol{\theta})$, where each clique $C_j$ is a subset of the variables in $\mathbf{z}$. Then the approximating joint distribution for $\mathcal{M}$ satisfies the same Markov network factorisation:*

$$Q_{\mathbf{z}}(\mathbf{z}) = \tilde{g} \prod_j \overline{\psi}_j(C_j)$$

*where $\overline{\psi}_j(C_j) = \exp\left\{ \langle \ln \psi_j(C_j, \boldsymbol{\theta}) \rangle_Q \right\}$ are new clique potentials obtained by averaging over $Q_{\boldsymbol{\theta}}(\boldsymbol{\theta})$, and $\tilde{g}$ is a normalisation constant. Furthermore, the expectations under the approximating posterior $Q_{\mathbf{x}}(\mathbf{x})$ required for the VE Step can be obtained by applying the junction tree algorithm.*

**Corollary 2: Conjugate-Exponential Markov Networks.** *Let $\mathcal{M}$ be a conjugate-exponential Markov network over the variables in $\mathbf{z}$. Then the approximating joint distribution for $\mathcal{M}$ is given by $Q_{\mathbf{z}}(\mathbf{z}) = \tilde{g} \prod_j \psi_j(C_j, \tilde{\boldsymbol{\theta}})$, where the clique potentials have exactly the same form as those in the original model but with natural parameters $\phi(\tilde{\boldsymbol{\theta}}) = \overline{\phi}(\boldsymbol{\theta})$.*

For conjugate-exponential models in which belief propagation and the junction tree algorithm over hidden variables is intractable further applications of Jensen's inequality can yield tractable factorisations in the usual way [7].

In the following section we derive a variational Bayesian treatment of linear-Gaussian state-space models. This serves two purposes. First, it will illustrate an application of Theorem 1. Second, linear-Gaussian state-space models are the cornerstone of stochastic filtering, prediction and control. A variational Bayesian treatment of these models provides a novel way to learn their structure, i.e. to identify the optimal dimensionality of their state-space.

## 5   State-space models

In state-space models (SSMs), a sequence of $D$-dimensional real-valued observation vectors $\{\mathbf{y}_1, \dots, \mathbf{y}_T\}$, denoted $\mathbf{y}_{1:T}$, is modeled by assuming that at each time step $t$, $\mathbf{y}_t$ was generated from a $K$-dimensional real-valued hidden state variable $\mathbf{x}_t$, and that the sequence of $\mathbf{x}$'s define a first-order Markov process. The joint probability of a sequence of states and observations is therefore given by (Figure 1):

$$P(\mathbf{x}_{1:T}, \mathbf{y}_{1:T}) = P(\mathbf{x}_1) P(\mathbf{y}_1|\mathbf{x}_1) \prod_{t=2}^{T} P(\mathbf{x}_t|\mathbf{x}_{t-1}) P(\mathbf{y}_t|\mathbf{x}_t).$$

We focus on the case where both the transition and output functions are linear and time-invariant and the distribution of the state and observation noise variables is Gaussian. This model is the linear-Gaussian state-space model:

$$\mathbf{x}_t = A\mathbf{x}_{t-1} + \mathbf{w}_t, \qquad \mathbf{y}_t = C\mathbf{x}_t + \mathbf{v}_t$$

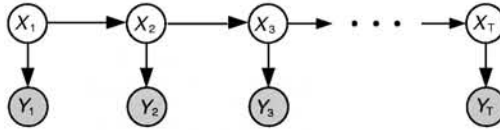

Figure 1: Belief network representation of a state-space model.

where $A$ and $C$ are the state transition and emission matrices and $\mathbf{w}_t$ and $\mathbf{v}_t$ are state and output noise. It is straightforward to generalise this to a linear system driven by some observed inputs, $\mathbf{u}_t$. A Bayesian analysis of state-space models using MCMC methods can be found in [4].

The complete data likelihood for state-space models is Gaussian, which falls within the class of exponential family distributions. In order to derive a variational Bayesian algorithm by applying the results in the previous section we now turn to defining conjugate priors over the parameters.

**Priors.** Without loss of generality we can assume that $\mathbf{w}_t$ has covariance equal to the unit matrix. The remaining parameters of a linear-Gaussian state-space model are the matrices $A$ and $C$ and the covariance matrix of the output noise, $\mathbf{v}_t$, which we will call $R$ and assume to be diagonal, $R = \mathrm{diag}(\boldsymbol{\rho})^{-1}$, where $\rho_i$ are the *precisions* (inverse variances) associated with each output.

Each row vector of the $A$ matrix, denoted $\mathbf{a}_i^\top$, is given a zero mean Gaussian prior with inverse covariance matrix equal to $\mathrm{diag}(\boldsymbol{\alpha})$. Each row vector of $C$, $\mathbf{c}_i^\top$, is given a zero-mean Gaussian prior with precision matrix equal to $\mathrm{diag}(\rho_i\boldsymbol{\beta})$. The dependence of the precision of $\mathbf{c}_i^\top$ on the noise output precision $\rho_i$ is motivated by conjugacy. Intuitively, this prior links the scale of the signal and noise.

The prior over the output noise covariance matrix, $R$, is defined through the precision vector, $\boldsymbol{\rho}$, which for conjugacy is assumed to be Gamma distributed[3] with hyperparameters $a$ and $b$: $P(\boldsymbol{\rho}\,|a,b) = \prod_{i=1}^{D} \frac{b^a}{\Gamma(a)}\rho_i^{a-1}\exp\{-b\rho_i\}$. Here, $\boldsymbol{\alpha}, \boldsymbol{\beta}$ are hyperparameters that we can optimise to do automatic relevance determination (ARD) of hidden states, thus inferring the structure of the SSM.

**Variational Bayesian learning for SSMs**

Since $A$, $C$, $\boldsymbol{\rho}$ and $\mathbf{x}_{1:T}$ are all unknown, given a sequence of observations $\mathbf{y}_{1:T}$, an exact Bayesian treatment of SSMs would require computing marginals of the posterior $P(A, C, \boldsymbol{\rho}, \mathbf{x}_{1:T}|\mathbf{y}_{1:T})$. This posterior contains interaction terms up to *fifth order* (for example, between elements of $C$, $\mathbf{x}$ and $\boldsymbol{\rho}$), and is not analytically manageable. However, since the model is conjugate-exponential we can apply Theorem 1 to derive a variational EM algorithm for state-space models analogous to the maximum-likelihood EM algorithm [10]. Moreover, since SSMs are singly connected belief networks Corollary 1 tells us that we can make use of belief propagation, which in the case of SSMs is known as the *Kalman smoother*.

Writing out the expression for $\log P(A, C, \boldsymbol{\rho}, \mathbf{x}_{1:T}, \mathbf{y}_{1:T})$, one sees that it contains interaction terms between $\boldsymbol{\rho}$ and $C$, but none between $A$ and either $\boldsymbol{\rho}$ or $C$. This observation implies a further factorisation, $Q(A, C, \boldsymbol{\rho}) = Q(A)Q(C, \boldsymbol{\rho})$, which falls out of the initial factorisation and the conditional independencies of the model.

Starting from some arbitrary distribution over the hidden variables, the VM step obtained by applying Theorem 1 computes the expected natural parameters of $Q_{\boldsymbol{\theta}}(\boldsymbol{\theta})$, where $\boldsymbol{\theta} = (A, C, \boldsymbol{\rho})$.

We proceed to solve for $Q(A)$. We know from Theorem 1 that $Q(A)$ is multivariate Gaussian, like the prior, so we only need to compute its mean and covariance. $A$ has mean $S^\top(\text{diag}(\boldsymbol{\alpha}) + W)^{-1}$ and each row of $A$ has covariance $(\text{diag}(\boldsymbol{\alpha}) + W)^{-1}$, where $S = \sum_{t=2}^T \langle \mathbf{x}_{t-1}\mathbf{x}_t^\top \rangle$, $W = \sum_{t=1}^{T-1} \langle \mathbf{x}_t\mathbf{x}_t^\top \rangle$, and $\langle . \rangle$ denotes averaging w.r.t. the $Q(\mathbf{x}_{1:T})$ distribution.

$Q(C, \boldsymbol{\rho})$ is also of the same form as the prior. $Q(\boldsymbol{\rho})$ is a product of Gamma densities $Q(\rho_i) = \mathcal{G}(\rho_i; \tilde{a}, \tilde{b}_i)$ where $\tilde{a} = a + \frac{T}{2}$, $\tilde{b}_i = b + \frac{1}{2}g_i$, $g_i = \sum_{t=1}^T y_{ti}^2 - U_i(\text{diag}(\boldsymbol{\beta}) + W')^{-1}U_i^\top$, $U_i = \sum_{t=1}^T y_{ti}\langle \mathbf{x}_t^\top \rangle$ and $W' = W + \langle \mathbf{x}_T\mathbf{x}_T^\top \rangle$. Given $\boldsymbol{\rho}$, each row of $C$ is Gaussian with covariance $\text{Cov}(\mathbf{c}_i) = (\text{diag}(\boldsymbol{\beta}) + W')^{-1}/\rho_i$ and mean $\bar{\mathbf{c}}_i = \rho_i U_i \text{Cov}(\mathbf{c}_i)$. Note that $S$, $W$ and $U_i$ are the expected complete data sufficient statistics $\bar{\mathbf{u}}$ mentioned in Theorem 1(a). Using the parameter distributions the hyperparameters can also be optimised.[4]

We now turn to the VE step: computing $Q(\mathbf{x}_{1:T})$. Since the model is a conjugate-exponential singly-connected belief network, we can use belief propagation (Corollary 1). For SSMs this corresponds to the Kalman smoothing algorithm, where every appearance of the natural parameters of the model is replaced with the following corresponding expectations under the $Q$ distribution: $\langle \rho_i \mathbf{c}_i \rangle$, $\langle \rho_i \mathbf{c}_i \mathbf{c}_i^\top \rangle$, $\langle A \rangle$, $\langle A^\top A \rangle$. Details can be found in [2].

Like for PCA [3], independent components analysis [1], and mixtures of factor analysers [5], the variational Bayesian algorithm for state-space models can be used to learn the structure of the model as well as average over parameters. Specifically, using $\mathcal{F}$ it is possible to compare models with different state-space sizes and optimise the dimensionality of the state-space, as we demonstrate in the following section.

## 6  Results

**Experiment 1:** The goal of this experiment was to see if the variational method could infer the structure of a variety of state space models by optimising over $\boldsymbol{\alpha}$ and $\boldsymbol{\beta}$. We generated a 200-step time series of 10-dimensional data from three models:[5] (a) a factor analyser (i.e. an SSM with $A = 0$) with 3 factors (static state variables); (b) an SSM with 3 dynamical interacting state variables, i.e. $A \neq 0$; (c) an SSM with 3 interacting dynamical and 1 static state variables. The variational Bayesian method correctly inferred the structure of each model in 2-3 minutes of CPU time on a 500 MHz Pentium III (Fig. 2 (a)–(c)).

**Experiment 2:** We explored the effect of data set size on complexity of the recovered structure. 10-dim time series were generated from a 6 state-variable SSM. On reducing the length of the time series from 400 to 10 steps the recovered structure became progressively less complex (Fig. 2(d)–(j)), to a 1-variable static model (j). This result agrees with the Bayesian perspective that the complexity of the model should reflect the data support.

**Experiment 3 (Steel plant):** 38 sensors (temperatures, pressures, etc) were sampled at 2 Hz from a continuous casting process for 150 seconds. These sensors covaried and were temporally correlated, suggesting a state-space model could capture some of its structure. The variational algorithm inferred that 16 state variables were required, of which 14 emitted outputs. While we do not know whether this is reasonable structure we plan to explore this as well as other real data sets.

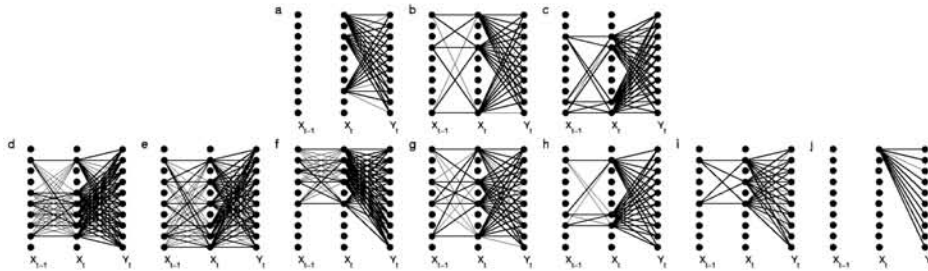

Figure 2: The elements of the $A$ and $C$ matrices after learning are displayed graphically. A link is drawn from node $k$ in $\mathbf{x}_{t-1}$ to node $l$ in $\mathbf{x}_t$ iff $\frac{1}{\alpha_k} > \epsilon$, and either $\frac{1}{\beta_l} > \epsilon$ or $\frac{1}{\alpha_l} > \epsilon$, for a small threshold $\epsilon$. Similarly links are drawn from node $k$ of $\mathbf{x}_t$ to $\mathbf{y}_t$ if $\frac{1}{\beta_k} > \epsilon$. Therefore the graph shows the links that take part in the dynamics and the output.

## 7  Conclusions

We have derived a general variational Bayesian learning algorithm for models in the conjugate-exponential family. There are a large number of interesting models that fall in this family, and the results in this paper should allow an almost automated protocol for implementing a variational Bayesian treatment of these models.

We have given one example of such an implementation, state-space models, and shown that the VB algorithm can be used to rapidly infer the hidden state dimensionality. Using the theory laid out in this paper it is straightforward to generalise the algorithm to mixtures of SSMs, switching SSMs, etc.

For conjugate-exponential models, integrating both belief propagation and the junction tree algorithm into the variational Bayesian framework simply amounts to computing expectations of the natural parameters. Moreover, the variational Bayesian algorithm contains EM as a special case. We believe this paper provides the foundations for a general algorithm for variational Bayesian learning in graphical models.

## Footnotes

[1]Models whose complete-data likelihood is not in the exponential family (such as ICA with the logistic nonlinearity, or sigmoid belief networks) can often be approximated by models in the exponential family with additional hidden variables.

[2]A tutorial on belief networks and Markov networks can be found in [9].

[3]More generally, if we let $R$ be a full covariance matrix for conjugacy we would give its inverse $V = R^{-1}$ a Wishart distribution: $P(V|\nu, S) \propto |V|^{(\nu-D-1)/2}\exp\left\{-\frac{1}{2}\mathrm{tr}\,VS^{-1}\right\}$, where tr is the matrix trace operator.

[4]The ARD hyperparameters become $\alpha_k = \frac{K}{\langle A^\top A \rangle_{kk}}$, and $\beta_k = \frac{D}{\langle C^\top \text{diag}(\boldsymbol{\rho})C \rangle_{kk}}$. The hyperparameters $a$ and $b$ solve the fixed point equations $\psi(a) = \ln b + \frac{1}{D}\sum_{i=1}^D \langle \ln \rho_i \rangle$, and $\frac{1}{b} = \frac{1}{aD}\sum_{i=1}^D \langle \rho_i \rangle$, where $\psi(w) = \frac{\partial}{\partial w} \ln \Gamma(w)$ is the *digamma* function.

[5]Parameters were chosen as follows: $R = I$, and elements of $C$ sampled from $\sim \text{Unif}(-5, 5)$, and $A$ chosen with eigen-values in $[0.5, 0.9]$.

## References

[1] H. Attias. A variational Bayesian framework for graphical models. In *Advances in Neural Information Processing Systems 12*. MIT Press, Cambridge, MA, 2000.

[2] M.J. Beal and Z. Ghahramani. The variational Kalman smoother. Technical report, Gatsby Computational Neuroscience Unit, University College London, 2000.

[3] C.M. Bishop. Variational PCA. In *Proc. Ninth ICANN*, 1999.

[4] S. Früwirth-Schnatter. Bayesian model discrimination and Bayes factors for linear Gaussian state space models. *J. Royal. Stat. Soc. B*, 57:237–246, 1995.

[5] Z. Ghahramani and M.J. Beal. Variational inference for Bayesian mixtures of factor analysers. In *Adv. Neur. Inf. Proc. Sys. 12*. MIT Press, Cambridge, MA, 2000.

[6] G.E. Hinton and D. van Camp. Keeping neural networks simple by minimizing the description length of the weights. In *Sixth ACM Conference on Computational Learning Theory, Santa Cruz*, 1993.

[7] M.I. Jordan, Z. Ghahramani, T.S. Jaakkola, and L.K Saul. An introduction to variational methods in graphical models. *Machine Learning*, 37:183–233, 1999.

[8] D.J.C. MacKay. Ensemble learning for hidden Markov models. Technical report, Cavendish Laboratory, University of Cambridge, 1997.

[9] J. Pearl. *Probabilistic Reasoning in Intelligent Systems: Networks of Plausible Inference*. Morgan Kaufmann, San Mateo, CA, 1988.

[10] R. H. Shumway and D. S. Stoffer. An approach to time series smoothing and forecasting using the EM algorithm. *J. Time Series Analysis*, 3(4):253–264, 1982.

[11] S. Waterhouse, D.J.C. Mackay, and T. Robinson. Bayesian methods for mixtures of experts. In *Adv. Neur. Inf. Proc. Sys. 7*. MIT Press, 1995.
